# Orientation, Scale, and Discontinuity as Emergent Properties of Illusory Contour Shape

**Karvel K. Thornber**
NEC Research Institute
4 Independence Way
Princeton, NJ 08540

**Lance R. Williams**
Dept. of Computer Science
University of New Mexico
Albuquerque, NM 87131

## Abstract

A recent neural model of illusory contour formation is based on a distribution of natural shapes traced by particles moving with constant speed in directions given by Brownian motions. The input to that model consists of pairs of position and direction constraints and the output consists of the distribution of contours joining all such pairs. In general, these contours will not be closed and their distribution will not be scale-invariant. In this paper, we show how to compute a scale-invariant distribution of closed contours given position constraints alone and use this result to explain a well known illusory contour effect.

## 1  INTRODUCTION

It has been proposed by Mumford[3] that the distribution of illusory contour shapes can be modeled by particles travelling with constant speed in directions given by Brownian motions. More recently, Williams and Jacobs[7, 8] introduced the notion of a *stochastic completion field*, the distribution of particle trajectories joining pairs of position and direction constraints, and showed how it could be computed in a local parallel network. They argued that the mode, magnitude and variance of the completion field are related to the observed shape, salience, and sharpness of illusory contours.

Unfortunately, the Williams and Jacobs model, as described, has some shortcomings. Recent psychophysics suggests that contour salience is greatly enhanced by closure[2]. Yet, in general, the distribution computed by the Williams and Jacobs model does not consist of closed contours. Nor is it scale-invariant—doubling the distances between the constraints does not produce a comparable completion field of

double the size without a corresponding doubling of the particle's speeds. However, the Williams and Jacobs model contains no intrinsic mechanism for speed selection. The speeds (like the directions) must be specified *a priori*. In this paper, we show how to compute a scale-invariant distribution of closed contours given position constraints alone.

## 2 TECHNICAL DETAILS

### 2.1 SHAPE DISTRIBUTION

Consistent with our earlier work[5, 6], in this paper we do not use the same distribution described by Mumford[3] but instead assume a distribution of completion shapes consisting of straight-line base-trajectories modified by random impulses drawn from a mixture of two limiting distributions. The first distribution consists of weak but frequently acting impulses (we call this the Gaussian-limit). The distribution of these weak impulses has zero mean and variance equal to $\sigma_g^2$. The weak impulses act at Poisson times with rate $R_g$. The second distribution consists of strong but infrequently acting impulses (we call this the Poisson-limit). Here, the magnitude of the random impulses is Gaussian distributed with zero mean. However, the variance is equal to $\sigma_p^2$ (where $\sigma_p^2 >> \sigma_g^2$). The strong impulses act at Poisson times with rate $R_p << R_g$. Particles decay with half-life equal to a parameter $\tau$. The effect is that particles tend to travel in smooth, short paths punctuated by occasional orientation discontinuities. See [5, 6].

### 2.2 EIGENSOURCES

Let $i$ and $j$ be position and velocity constraints, $(\mathbf{x}_i, \dot{\mathbf{x}}_i)$ and $(\mathbf{x}_j, \dot{\mathbf{x}}_j)$. Then $P(j|i)$ is the conditional probability that a particle beginning at $i$ will reach $j$. Note that these transition probabilities are not symmetric, i.e., $P(j|i) \neq P(i|j)$. However, by time-reversal symmetry, $P(j|i) = P(\bar{i}|\bar{j})$ where $\bar{i} = (\mathbf{x}_i, -\dot{\mathbf{x}}_i)$ and $\bar{j} = (\mathbf{x}_j, -\dot{\mathbf{x}}_j)$.

Given only the matrix of transition probabilities, $\mathbf{P}$, we would like to compute the relative number of closed contours satisfying a given position and velocity constraint. We begin by noting that, due to their randomness, only increasingly smaller and smaller fractions of contours are likely to satisfy increasing numbers of constraints. Suppose we let $s_i^{(1)}$ contours start at $\mathbf{x}_i$ with $\dot{\mathbf{x}}_i$. Then

$$s_j^{(2)} = \sum_i P(j|i)s_i^{(1)}$$

is the relative number of contours through $\mathbf{x}_j$ with $\dot{\mathbf{x}}_j$, i.e., which satisfy two constraints. In general,

$$s_j^{(n+1)} = \sum_i P(j|i)s_i^{(n)}$$

Now suppose we compute the eigenvector,

$$\lambda s_j = \sum_i P(j|i)s_i$$

with largest, real positive eigenvalue, and take $s_i^{(1)} = s_i$. Then clearly $s_i^{(n+1)} = \lambda^n s_i$. This implies that as the number of constraints satisfied increases by one, the number of contours remaining in the sample of interest decreases by $\lambda$. However, the ratios of the $s_i$ remain invariant. Letting $n$ pass to infinity, we see that the $s_i$ are just the relative number of contours through $i$. To summarize, having started with all possible contours, we are now left with only those bridging pairs of constraints at all past-times. By solving $\lambda \mathbf{s} = \mathbf{P}\mathbf{s}$ for $\mathbf{s}$ we know their relative numbers. We refer to the components of $\mathbf{s}$ as the *eigensources* of the stochastic completion field.

## 2.3  STOCHASTIC COMPLETION FIELDS

Note that the eigensources alone do not represent a distribution of closed contours. In fact, the majority of contours contributing to s will not satisfy a single additional constraint. However, the following recurrence equation gives the number of contours which begin at constraint $i$ and end at constraint $j$ and satisfy $n-1$ intermediate constraints

$$P^{(n+1)}(j\,|\,i) = \sum_k P(j\,|\,k)P^{(n)}(k\,|\,i)$$

where $P^{(1)}(j\,|\,i) = P(j\,|\,i)$. Given the above recurrence equation, we can define an expression for the relative number of contours of any length which begin and end at constraint $i$:

$$c_i = \lim_{n\to\infty} P^{(n)}(i\,|\,i) / \sum_j P^{(n)}(j\,|\,j)$$

Using a result from the theory of positive matrices[1], it is possible to show that the above expression is simply

$$c_i = s_i \bar{s}_i / \sum_j s_j \bar{s}_j$$

where s and $\bar{\text{s}}$ are the right and left eigenvectors of **P** with largest positive real eigenvalue, i.e., $\lambda$s = **P**s and $\lambda\bar{\text{s}}$ = $\mathbf{P^T}\bar{\text{s}}$. Because of the time-reversal symmetry of **P**, the right and left eigenvectors are related by a permutation which exchanges opposite directions, i.e., $\bar{s}_i = s_{\bar{i}}$.

Finally, given s and $\bar{\text{s}}$, it is possible to compute the relative number of closed contours through an *arbitrary* position and velocity in the plane, i.e., to compute the stochastic completion field. If $\eta = (\mathbf{x}, \dot{\mathbf{x}})$ is an arbitrary position and velocity in the plane, then

$$C(\eta) = \tfrac{1}{\lambda \mathbf{s^T}\bar{\mathbf{s}}} \sum_i P(\eta\,|\,i)s_i \cdot \sum_j P(j\,|\,\eta)\bar{s}_j$$

gives the relative probability that a closed contour will pass through $\eta$. Note, that this is a natural generalization of the Williams and Jacobs[7] factorization of the completion field into the product of source and sink fields.

## 2.4  SCALE-INVARIANCE

Under the restriction that particles have constant speed, the transition probability matrix, **P**, becomes block-diagonal. Each block corresponds to a different possible speed, $\gamma$. Since the components of any given eigenvector will be confined to a single block, we can consider **P** to be a function of $\gamma$ and solve:

$$\lambda(\gamma)\,\mathbf{s}(\gamma) \;=\; \mathbf{P}(\gamma)\mathbf{s}(\gamma)$$

Let $\lambda_{max}(\gamma)$ be the largest positive real eigenvalue of $\mathbf{P}(\gamma)$ and let $\gamma_{max}$ be the speed where $\lambda_{max}(\gamma)$ is maximized. Then $\mathbf{s}_{max}(\gamma_{max})$, i.e., the eigenvector of $\mathbf{P}(\gamma_{max})$ associated with $\lambda_{max}(\gamma_{max})$, is the limiting distribution over all spatial scales.

## 3  EXPERIMENTS

### 3.1  EIGHT POINT CIRCLE

Given eight points spaced uniformly around the perimeter of a circle of diameter, $d = 16$, we would like to find the distribution of directions through each point and the corresponding completion field (Figure 1 (left)). Neither the order of traversal, directions, i.e., $\dot{\mathbf{x}}_i / |\dot{\mathbf{x}}_i|$, or speed, i.e., $\gamma = |\dot{\mathbf{x}}_i|$, are specified *a priori*. In all of our experiments, we sample direction at $5°$ intervals. Consequently, there are 72 discrete directions and 576 position-direction pairs, i.e., $\mathbf{P}(\gamma)$ is of size $576 \times 576$.[1]

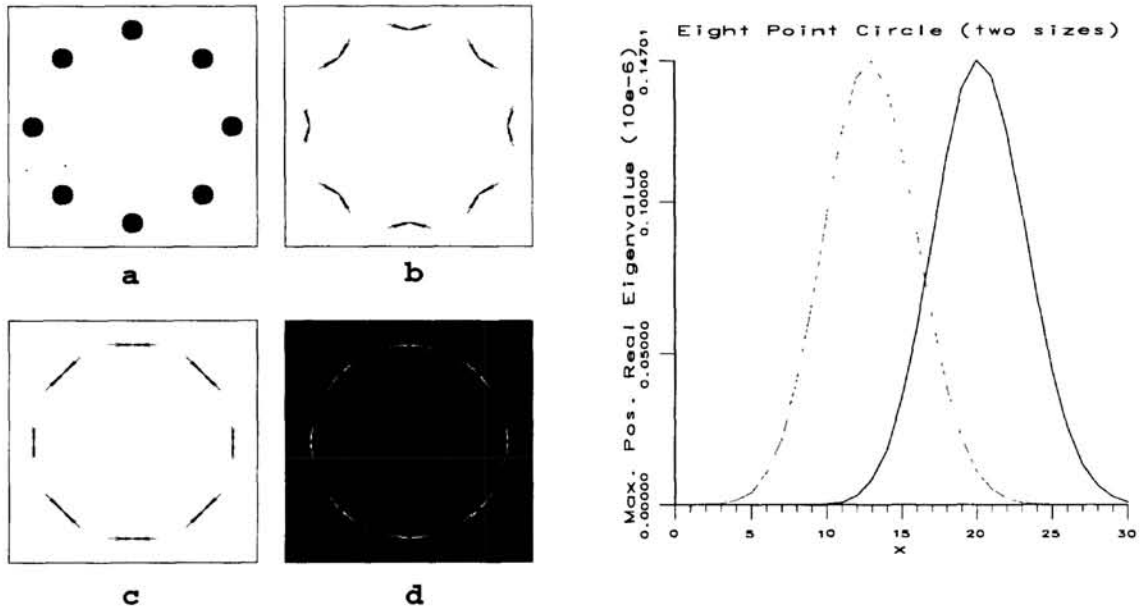

Figure 1: Left: **(a)** The eight position constraints. Neither the order of traversal, directions, or speed are specified *a priori*. **(b)** The eigenvector, $s_{max}(\gamma_{max})$ represents the limiting distribution over all spatial scales. **(c)** The product of $s_{max}(\gamma_{max})$ and $\bar{s}_{max}(\gamma_{max})$. Orientations tangent to the circle dominate the distribution of closed contours. **(d)** The stochastic completion field, $C$, due to $s_{max}(\gamma_{max})$. Right: Plot of magnitude of maximum positive real eigenvalue, $\lambda_{max}$, vs. $\log_{1.1}(1/\gamma)$ for eight point circle with $d = 16.0$ (solid) and $d = 32.0$ (dashed).

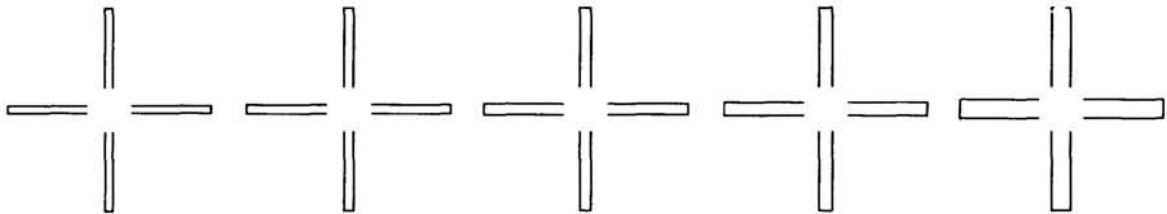

Figure 2: Observers report that as the width of the arms increases, the shape of the illusory contour changes from a circle to a square[4].

First, we evaluated $\lambda_{max}(\gamma)$ over the velocity interval $[1.1^{-1}, 1.1^{-30}]$ using standard numerical routines and plotted the magnitude of the largest, real positive eigenvalue, $\lambda_{max}$ vs. $\log_{1.1}(1/\gamma)$. The function reaches its maximum value at $\gamma_{max} \approx 1.1^{-20}$. Consequently, the eigenvector, $s_{max}(1.1^{-20})$ represents the limiting distribution over all spatial scales (Figure 1 (right)).

Next, we scaled the test Figure by a factor of two, i.e., $d' = 32.0$ and plotted $\lambda'_{max}(\gamma)$ over the same interval (Figure 1 (right)). We observe that $\lambda'_{max}(1.1^{-x+7}) \approx \lambda_{max}(1.1^{-x})$, i.e., when plotted using a logarithmic $x$-axis, the functions are identical except for a translation. It follows that $\gamma'_{max} \approx \log_{1.1} 7 \times \gamma_{max} \approx 2.0 \times \gamma_{max}$. This confirms the scale-invariance of the system—doubling the size of the Figure results in a doubling of the selected speed.

## 3.2   KOFFKA CROSS

The Koffka Cross stimulus (Figure 2) has two basic degrees of freedom which we call diameter (i.e., $d$) and arm width (i.e., $w$) (Figure 3 (a)). We are interested in how

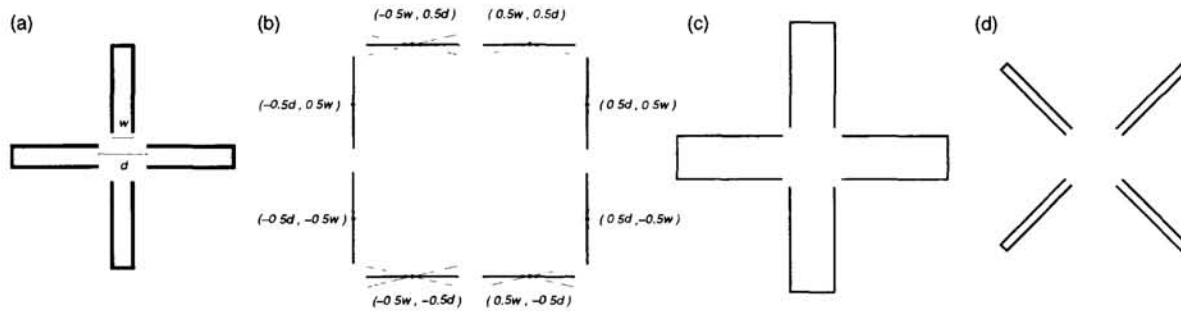

Figure 3: **(a)** Koffka Cross showing diameter, $d$, and width, $w$. **(b)** Orientation and position constraints in terms of $d$ and $w$. The normal orientation at each endpoint is indicated by the solid lines while the dashed lines represent plus or minus one standard deviation (i.e., 12.8°) of the Gaussian weighting function. **(c)** Typically perceived as square. **(d)** Typically perceived as circle. The positions of the line endpoints is the same.

the stochastic completion field changes as these parameters are varied. Observers report that as the width of the arms increases, the shape of the illusory contour changes from a circle to a square[4]. The endpoints of the lines comprising the Koffka Cross can be used to define a set of position and orientation constraints (Figure 3 (b)). The position constraints are specified in terms of the parameters, $d$ and $w$. The orientation constraints take the form of a Gaussian weighting function which assigns higher probabilities to contours passing through the endpoints with orientations normal to the lines.[2] The prior probabilities assigned to each position-direction pair by the Gaussian weighting function form a diagonal matrix, $\mathbf{D}$:

$$\lambda(\gamma)\,\mathbf{s}(\gamma) \;=\; \mathbf{D}^{\frac{1}{2}}\mathbf{P}(\gamma)\mathbf{D}^{\frac{1}{2}}\mathbf{s}(\gamma) \;=\; \mathbf{Q}(\gamma)\mathbf{s}(\gamma)$$

where $\mathbf{P}(\gamma)$ is the transition probability matrix for the random process at scale $\gamma$, $\lambda(\gamma)$ is an eigenvalue of $\mathbf{Q}(\gamma)$, and $\mathbf{s}(\gamma)$ is the corresponding eigenvector. Let $\lambda_{max}(\gamma)$ be the largest positive real eigenvalue of $\mathbf{Q}(\gamma)$ and let $\gamma_{max}$ be the scale where $\lambda_{max}(\gamma)$ is maximized. Then $\mathbf{s}_{max}(\gamma_{max})$, i.e., the eigenvector of $\mathbf{Q}(\gamma_{max})$ associated with $\lambda_{max}(\gamma_{max})$, is the limiting distribution over all spatial scales.

First, we used a Koffka Cross where $d = 2.0$ and $w = 0.5$ and evaluated $\lambda_{max}(\gamma)$ over the velocity interval $[8.0 \times 1.1^{-1}, 8.0 \times 1.1^{-80}]$ using standard numerical routines.[3] The function reaches its maximum value at $\gamma_{max} \approx 8.0 \times 1.1^{-62}$ (Figure 4 (left)). Observe that the completion field due to the eigenvector, $\mathbf{s}_{max}(8.0 \times 1.1^{-62})$, is dominated by contours of a predominantly circular shape (Figure 4 (right)). We then uniformly scaled the Koffka Cross Figure by a factor of two, i.e., $d' = 4.0$ and

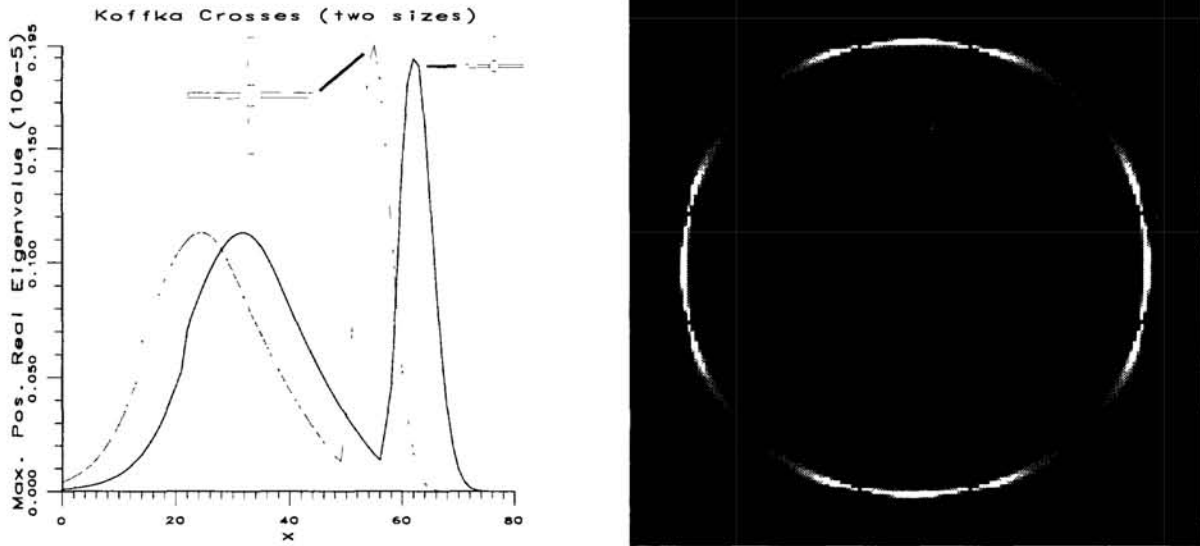

Figure 4: Left: Plot of magnitude of maximum positive real eigenvalue, $\lambda_{max}$, vs. $\log_{1.1}(1/\gamma)$ for Koffka Crosses with $d = 2.0$ and $w = 0.5$ (solid) and $d = 4.0$ and $w = 1.0$ (dashed). Right: The completion field due to the eigenvector, $s_{max}(8.0 \times 1.1^{-62})$.

$w' = 1.0$ and plotted $\lambda'_{max}(\gamma)$ over the same interval (Figure 4 (left)). Observe that $\lambda'_{max}(8.0 \times 1.1^{-x+7}) \approx \lambda_{max}(8.0 \times 1.1^{-x})$. As before, this confirms the scale-invariance of the system.

Next, we studied how the relative magnitudes of the local maxima of $\lambda_{max}(\gamma)$ change as the parameter $w$ is varied. We begin with a Koffka Cross where $d = 2.0$ and $w = 0.5$ and observe that $\lambda_{max}(\gamma)$ has two local maxima (Figure 5 (left)). We refer to the larger of these maxima as $\gamma_{circle}$. As previously noted, this maximum is located at approximately $8.0 \times 1.1^{-62}$. The second maximum is located at approximately $8.0 \times 1.1^{-32}$. When the completion field due to the eigenvector, $s_{max}(8.0 \times 1.1^{-32})$, is rendered, we observe that the distribution is dominated by contours of predominantly square shape (Figure 5(a)). For this reason, we refer to this local maximum as $\gamma_{square}$. Now consider a Koffka Cross where the widths of the arms are doubled but the diameter remains the same, i.e., $d' = 2.0$ and $w' = 1.0$. We observe that $\lambda'_{max}(\gamma)$ still has two local maxima, one at approximately $8.0 \times 1.1^{-63}$ and a second at approximately $8.0 \times 1.1^{-29}$ (Figure 5 (left)). When we render the completion fields due to the eigenvectors, $s'_{max}(8.0 \times 1.1^{-63})$ and $s'_{max}(8.0 \times 1.1^{-29})$, we find that the completion fields have the same general character as before—the contours associated with the smaller spatial scale (i.e., lower speed) are approximately circular and those associated with the larger spatial scale (i.e., higher speed) are approximately square (Figure 5 (d) and (c)). Accordingly, we refer to the locations of the respective local maxima as $\gamma'_{circle}$ and $\gamma'_{square}$. However, what is most interesting is that the relative magnitudes of the local maxima have reversed. Whereas we previously observed that $\lambda_{max}(\gamma_{circle}) > \lambda_{max}(\gamma_{square})$, we now observe that $\lambda'_{max}(\gamma'_{square}) > \lambda'_{max}(\gamma'_{circle})$. Therefore, the completion field due to the eigenvector, $s'_{max}(\gamma'_{square})$ [not $s'_{max}(\gamma'_{circle})$!] represents the limiting distribution over all spatial scales. This is consistent with the transition from circle to square reported by human observers when the widths of the arms of the Koffka Cross are increased.

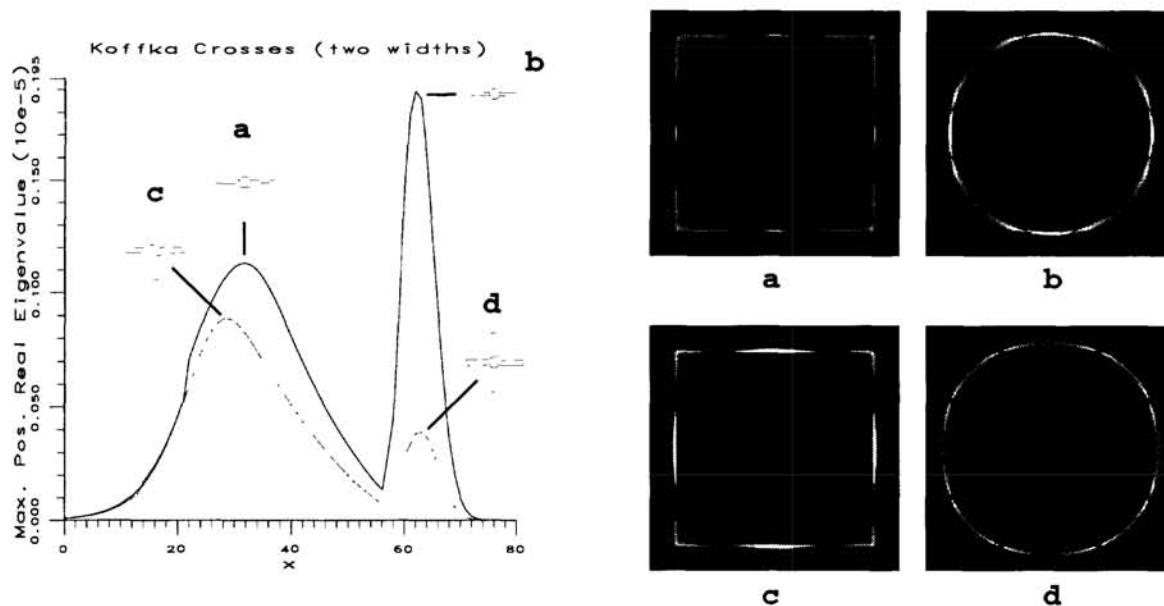

Figure 5: Plot of magnitude of maximum positive real eigenvalue, $\lambda_{max}$, vs. $\log_{1.1}(1/\gamma)$ for Koffka Crosses with $d = 2.0$ and $w = 0.5$ (solid) and $d = 2.0$ and $w = 1.0$ (dashed). Stochastic completion fields for Koffka Cross due to **(a)** $\mathbf{s}_{max}(\gamma_{square})$ is a local optimum for $w = 0.5$ **(b)** $\mathbf{s}_{max}(\gamma_{circle})$ is the global optimum for $w = 0.5$ **(c)** $\mathbf{s}'_{max}(\gamma'_{square})$ is the global optimum for $w = 1.0$ **(d)** $\mathbf{s}'_{max}(\gamma'_{square})$ is a local optimum for $w = 1.0$. These results are consistent with the circle-to-square transition perceived by human subjects when the width of the arms of the Koffka Cross are increased.

# 4   CONCLUSION

We have improved upon a previous model of illusory contour formation by showing how to compute a scale-invariant distribution of closed contours given position constraints alone. We also used our model to explain a previously unexplained perceptual effect.

## Footnotes

[1]The parameters defining the distribution of completion shapes are $T = R_g \sigma_g^2 = 0.0005$ and $\tau = 9.5$. For simplicity, we assume the pure Gaussian-limit case described in [6].

[2]Observe that Figure 3 (c) is perceived as a square while Figure 3 (d) is perceived as a circle. Yet the positions of the line endpoints is the same. It follows that the orientations of the lines affect the percept. We have chosen to model this dependence through the use of a Gaussian weighting function which favors contours passing through the endpoints of the lines in the normal direction. It is possible to motivate this based on the statistics of natural scenes. The distribution of relative orientations at contour crossings is maximum at 90° and drops to nearly zero at 0° and 180°.

[3]The parameters defining the distribution of completion shapes were: $T = R_g\sigma_g^2 = 0.0005$, $\tau = 9.5$, $\xi_p = \sigma_p^2/T = 100.0$ and $R_p = 1.0 \times 10^{-8}$. As an anti-aliasing measure, the transition probabilities, $P(j \mid i)$, were averaged over initial conditions modeled as Gaussians of variance $\sigma_x^2 = \sigma_y^2 = 0.00024$ and $\sigma_\theta^2 = 0.0019$. See [6].

# References

[1] Horn, R.A., and C.R. Johnson, *Matrix Analysis*, Cambridge Univ. Press, p. 500, 1985.

[2] Kovacs, I. and B. Julesz, A Closed Curve is Much More than an Incomplete One: Effect of Closure in Figure-Ground Segmentation, *Proc. Natl. Acad. Sci. USA*, **90**, pp. 7495-7497, 1993.

[3] Mumford, D., Elastica and Computer Vision, *Algebraic Geometry and Its Applications*, Chandrajit Bajaj (ed.), Springer-Verlag, New York, 1994.

[4] Sambin, M., Angular Margins without Gradients, *Italian Journal of Psychology* **1**, pp. 355-361, 1974.

[5] Thornber, K.K. and L.R. Williams, Analytic Solution of Stochastic Completion Fields, *Biological Cybernetics* **75**, pp. 141-151, 1996.

[6] Thornber, K.K. and L.R. Williams, Characterizing the Distribution of Completion Shapes with Corners Using a Mixture of Random Processes, *Intl. Workshop on Energy Minimization Methods in Computer Vision*, Venice, Italy, 1997.

[7] Williams, L.R. and D.W. Jacobs, Stochastic Completion Fields: A Neural Model of Illusory Contour Shape and Salience, *Neural Computation* **9**(4), pp. 837-858, 1997.

[8] Williams, L.R. and D.W. Jacobs, Local Parallel Computation of Stochastic Completion Fields, *Neural Computation* **9**(4), pp. 859-881, 1997.
